# Classification of Electroencephalogram using Artificial Neural Networks

A C Tsoi*, D S C So*, A Sergejew**
*Department of Electrical Engineering
**Department of Psychiatry
University of Queensland
St Lucia, Queensland 4072
Australia

## Abstract

In this paper, we will consider the problem of classifying electroencephalogram (EEG) signals of normal subjects, and subjects suffering from psychiatric disorder, e.g., obsessive compulsive disorder, schizophrenia, using a class of artificial neural networks, viz., multi-layer perceptron. It is shown that the multilayer perceptron is capable of classifying unseen test EEG signals to a high degree of accuracy.

## 1  Introduction

The spontaneous electrical activity of the brain was first observed by Caton in 1875. Although considerable investigations on the electrical activity of the non-human brain have been undertaken, it was not until 1929 that a German neurologist Hans Berger first published studies on the electroencephalogram (EEG) recorded on the scalp of human. He lay the foundation of clinical and experimental applications of EEG between 1929 and 1938.

Since then EEG signals have been used in both clinical and experimental work to discover the state which the brain is in (see e.g., Herrmann, 1982, Kolb and Whishaw, 1990, Lindsay and Holmes, 1984). It has served as a direct indication of any brain activities. It is routinely being used in clinical diagnosis of epilepsy (see e.g., Basar, 1980; Cooper, 1980).

Despite advances in technology, the classification of EEG signals at present requires a trained personnel who either "eyeballs" the direct EEG recordings over time,

or studies the contour maps representing the potentials generated from the "raw" electrical signal (see e.g., Cooper, 1980). This is both a highly skillful job, as well as a laborious task for a neurologist. With the current advances in computers, a logical question to ask: can we use the computer to perform an automatic classification of EEG signals into different classes denoting the psychiatric states of the subjects?

This type of classification studies is not new. In fact, in the late 1960's there were a number of attempts in performing the automatic classification using discriminant analysis techniques. However, this work was largely abandoned as most researchers concluded that classification based on discriminant techniques does not generalise well, i.e., while it has very good classification accuracies in classifying the data which is used to train the automatic classification system, it may not have high accuracy in classifying the unseen data which are not used to train the system in the first instance.

Recently, a class of classification techniques, called artificial neural network (ANN), based on nonlinear models, has become very popular (see e.g., Touretzky, 1989, 1990, Lippmann et al, 1991). This type of networks claims to be inspired by biological neurons, and their many inter-connections. This type of artificial neural networks has limited pattern recognition capabilities. Among the many applications which have been applied so far are sonar signal classification (see e.g., Touretzky, 1989), handwritten character recognition (see .e.g., Touretzky, 1990), facial expression recognition (see e.g., Lippmann et al. 1991).

In this paper, we will investigate the possibility of using an ANN for EEG classifications. While it is possible to extract features from the time series using either time domain or frequency domain techniques, from some preliminary work, it is found that the time domain techniques give much better results.

The structure of this paper is as follows: In section 2, we will give a brief discussion on a popular class of ANNs, viz., multi-layer perceptrons (MLP). In section 3, we will discuss various feature extractions using time domain techniques. In section 4, we will present results in classifying a set of unseen EEG signals.

## 2   Multi-layer Perceptrons

Artificial neural network (ANN) consists of a number of artificial neurons interconnected together by synaptic weights to form a network (see e.g, Lippmann, 1987). Each neuron is modeled by the following mechanical model:

$$y = f(\sum_{i=1}^{n} w_i x_i + \theta) \tag{1}$$

where $y$ is the output of the neuron, $w_i, i = 1, 2, \ldots, n$ are the synaptic weights, $x_i, i = 1, 2 \ldots, n$ are the inputs, and $\theta$ is a threshold function. The nonlinear function $f(.)$ can be a sigmoid function, or a hyperbolic tangent function. An ANN is a network of inter-connected neurons by synapses (Hertz, Krogh and Palmer, 1991).

There are many possible ANN architectures (Hertz, Krogh, Palmer, 1991). A pop-

ular architecture is the multi-layer perceptron (MLP) (see e.g., Lippmann, 1987). In this class of ANN, signal travels only in a forward direction. Hence it is also known as a feedforward network. Mathematically, it can be described as follows:

$$y = f(Az + \theta_y) \qquad (2)$$
$$z = f(Bu + \theta_z) \qquad (3)$$

where $y$ is a $m \times 1$ vector, representing the output of the output layer neurons; $z$ is a $p \times 1$ vector, representing the outputs of the hidden layer neurons; $u$ is a $n \times 1$ vector, representing the input feature vector; $\theta_y$ is a $m \times 1$ vector, known as the threshold vector for the output layer neurons; $\theta_z$ is a $p \times 1$ vector, representing the threshold vector for the hidden layer neurons; $A$ and $B$ are matrices of $m \times p$ and $p \times n$ respectively. The matrices $A$, and $B$ are the synaptic weights connecting the hidden layer neuron to the output layer neuron; and the input layer neurons, and the hidden layer neurons respectively. For simplicity sake, we will assume the nonlinearity function to be a sigmoid function, i.e.,

$$f(\alpha) = \frac{1}{1 + e^{-\alpha}} \qquad (4)$$

The unknown parameters $A, B, \theta_y, \theta_z$ can be obtained by minimizing an error criterion:

$$J = \sum_{i=1}^{P} (d_i - y_i)^2 \qquad (5)$$

where $P$ is the total number of examplars, $d_i, i = 1, 2, \ldots, P$ are the desired outputs which we wish the MLP to learn.

By differentiating the error criterion J with respect to the unknown parameters, learning algorithms can be obtained.

The learning rules are as follows:

$$A^{new} = A^{old} + \eta \Lambda(y) e z^T \qquad (6)$$

where $A^{new}$ is the next estimate of the matrix $A$, $T$ denotes the transpose of a vector or a matrix. $\eta$ is a learning constant. $\Lambda(y)$ is a $m \times m$ diagonal matrix, whose diagonal elements are $f'(y_i), i = 1, 2, \ldots, m$. The vector $e$ is $m \times 1$, and it is given by $e^T = [(d_1 - y_1), (d_2 - y_2), \ldots, (d_m - y_m)]^T$.

The updating equation for the $B$ matrix is given by the following

$$B^{new} = B^{old} + \eta \Lambda(z) \delta u^T \qquad (7)$$

where $\delta$ is a $p \times 1$ vector, given by

$$\delta = A^T \Lambda(y)e \tag{8}$$

and the other parameters are as defined above.

The threshold vectors can be obtained as follows:

$$\theta_y^{new} = \theta_y^{old} + \eta\Lambda(y)e \tag{9}$$

and

$$\theta_z^{new} = \theta_z^{old} + \eta\Lambda(z)\delta \tag{10}$$

Thus it is observed that once a set of initial conditions for the unknown parameters are given, this algorithm will find a set of parameters which will converge to a value, representing possibly a local minimum of the error criterion.

## 3   Pre-processing of the EEG signal

A cursory glance at a typical EEG signal of a normal subject, or a psychiatrically ill subject would convince anyone that one cannot hope to distinguish the signal just from the raw data alone. Consequently, one would need to perform considerable feature extraction (data pre-processing) before classification can be made. There are two types of simple feature extraction techniques, viz., frequency domain and time domain (see e.g., Kay, 1988, Marple, 1987). In the frequency domain, one performs a fast Fourier transform (FFT) on the data. Often it is advantageous to modify the signal by a window function. This will reduce the sidelobe leakage (Kay, and Marple, 1981, Harris, 1978). it is possible to use the average spectrum, obtained by averaging the spectrum over a number of frames, as the input feature vector to the MLP.

In the time domain, one way to pre-process the data is to fit a parametric model to the underlying data. There are a number of parametric models, e.g., autoregressive (AR) model, an autoregressive moving average (ARMA) model (see e.g., Kay, 1988, Marple, 1987).

The autoregressive model can be described as follows:

$$s_t = \sum_{j=1}^{N} \alpha_j s_{t-j} + \epsilon_t \tag{11}$$

where $s_t$ is the signal at time $t$; $\epsilon_t$ is assumed to be a zero mean Gaussian variable with variance $\sigma^2$. The unknown parameters $\alpha_j, j = 1, 2, \ldots, N$ describe the spectrum of the signal. They can be obtained by using standard methods, e.g., Yule-Walker equations, or Levinson algorithm (Kay, 1988, Marple, 1987).

The autoregressive moving average (ARMA) model can be seen as a parsimonious model for an AR model with a large $N$. Hence, as long as we are not concerned

about the interpretation of the AR model obtained, there is little advantage to use the more complicated ARMA model. Subsequently, in this paper, we will only consider the AR models.

Once the AR parameters are determined, then they can be used as the input features to the MLP. It is known that the AR parametric model basically produces a smoothed spectral envelope (Kay, 1988, Marple, 1987). Thus, the model parameters of AR is another way to convey the spectral information to the MLP. This information is different in quality to that given by the FFT technique in that the FFT transforms both signal and noise alike, while the parametric models tend to favor the signal more and is more effective in suppressing the noise effect.

In some preliminary work, we find that the frequency domain extracted features do not give rise to good classification results using MLP. Henceforth we will consider only the AR parameters as input feature vectors.

## 4    Classification Results

In this section, we will summarise the results of the experiments in using the AR parametric method of feature extraction as input parameters to the MLP.

We obtained EEG data pertaining to normal subjects, subjects who have been diagnosed as suffering from severe obsessive compulsive disorder (OCD), and subjects who have been diagnosed as suffering from severe schizophrenia. Both the OCD and the schizophrenic subjects are under medication. The subjects are chosen so that their medication as well as their medical conditions are at a steady state, i.e., they have not changed over a long period of time. The diagnosis is made by a number of trained neurologists. The data files are chosen only if the diagnosis from the experts concur.

We use the standard 10-20 recording system (Cooper, 1980), i.e., there are 19 channels of EEG recording, each sampled at 128 Hz. The recording were obtained while the subject is at rest. Some data screening has been performed to screen out the segment of data which contains any artifact. In addition, the data is anti-aliased first by a low pass filter before being sampled. The sampled data is then low pass filtered at 30 Hz to get rid of any higher frequency components.

We have chosen one channel, viz., the $C_z$ channel (the channel which is the recording of the signal at the azimuth of the scalp). This channel can be assumed to be representative of the brain state from the overall EEG recording of the scalp. [1] This time series is employed for feature extraction purposes.

For time domain feature extraction, we first convert the time series into a zero mean one. Then a data frame of one second duration is chosen [2] as the basic time segmentation of the series. An AR model is fitted to this one second time frame to

extract a feature vector formed by the resulting AR coefficients.

An average feature vector is acquired from the first 250 seconds, as in practice, the first 250 seconds usually represent a state of calm in the patient, and therefore the EEG is less noisy. After the first 250 seconds, the patient may enter an unstable condition, such as breathing faster and muscle contraction which can introduce artifacts. We use an AR model of length between 8 to 15.

We have chosen 15 such data file to form our training data set. This consists of 5 data files from normal subjects, 5 from OCD subjects, and 5 from subjects suffering from schizophrenia.

In the time domain extracted feature vectors, we use a MLP with 8 input neurons, 15 hidden layer neurons, and 3 output neurons. The MLP's are trained accordingly. We use a learning gain of 0.01. Once trained, the network is used to classify unseen data files. These unseen data files were pre-classified by human experts. Thus the desired classification of the unseen data files are known. This can then be used to check the usefulness of the MLP in generalising to unseen data files.

The results [3] are shown in table 1.

The unseen data set consists of 6 normal subjects, 8 schizophrenic subjects, and 10 obsessive compulsive disorder subjects. It can be observed that the network correctly classifies all the normal cases, makes one mistake in classifying the schizophrena cases, and one mistake in classifying the OCD cases.

Also we have experimented on varying the number of hidden neurons. It is found that the classification accuracy does not vary much with the variation of hidden layer neurons from 15 to 50.

We have also applied the MLP on the frame by frame data, i.e., before they are being averaged over the 250 second interval. However, it is found that the classification results are not as good as the ones presented. We were puzzled by this result as intuitively, we would expect the frame by frame results to be better than the ones presented.

A plausible explanation for this puzzle is given as follows: the EEG data is in general quite noisy. In the frame by frame analysis, the features extracted may vary considerably over a short time interval, while in the approach taken here, the noise effect is smoothed out by the averaging process.

One may ask: why would the methods presented work at all? In traditional EEG analysis (Lindsay & Holmes, 1984), FFT technique is used to extract the frame by frame frequency responses. The averaged frequency response is then obtained over this interval. Traditionally only four dominant frequencies are observed, viz., the "alpha", "beta", "delta", and "theta" frequencies. It is a basic result in EEG research that these frequencies describe the underlying state of the subject. For example, it is known that the "alpha" wave indicates that the subject is at rest. An EEG technologist uses data in this form to assist in the diagnosis of the subject. On the other hand, it is relatively well known in signal processing literature (Kay,

| original classes | activation of normal | activation of schiz | activation of ocd | predicted classes |
|---|---|---|---|---|
| normal1 | 0.905 | 0.008 | 0.201 | normal |
| normal2 | 0.963 | 0.006 | 0.103 | normal |
| normal3 | 0.896 | 0.021 | 0.086 | normal |
| normal4 | 0.870 | 0.057 | 0.020 | normal |
| normal5 | 0.760 | 0.237 | 0.000 | normal |
| normal6 | 0.752 | 0.177 | 0.065 | normal |
| schiz1 | 0.000 | 0.981 | 0.042 | schiz |
| schiz2 | 0.000 | 0.941 | 0.163 | schiz |
| schiz3 | 0.002 | 0.845 | 0.050 | schiz |
| schiz4 | 0.015 | 0.989 | 0.004 | schiz |
| schiz5 | 0.000 | 0.932 | 0.061 | schiz |
| schiz6 | 0.377 | 0.695 | 0.014 | schiz |
| schiz7 | 0.062 | 0.898 | 0.000 | schiz |
| schiz8 | 0.006 | 0.086 | 0.921 | ocd |
| ocd1 | 0.017 | 0.134 | 0.922 | ocd |
| ocd2 | 0.027 | 0.007 | 0.940 | ocd |
| ocd3 | 0.000 | 0.033 | 0.993 | ocd |
| ocd4 | 0.000 | 0.014 | 0.997 | ocd |
| ocd5 | 0.015 | 0.138 | 0.889 | ocd |
| ocd6 | 0.000 | 0.150 | 0.946 | ocd |
| ocd7 | 0.002 | 0.034 | 0.985 | ocd |
| ocd8 | 0.006 | 0.960 | 0.003 | schiz |
| ocd9 | 0.045 | 0.005 | 0.940 | ocd |
| ocd10 | 0.085 | 0.046 | 0.585 | ocd |

Table 1: Classification of unseen EEG data files

1988, Marple, 1987) to view the AR model as indicative of the underlying frequency content of the signal. In fact, an 8th order AR model indicates that the signal can be considered to consist of 4 underlying frequencies. Thus, intuitively, the 8th order AR model averaged over the first 250 seconds represents the underlying dominant frequencies in the signal. Given this interpretation, it is not surprising that the results are so good. The features extracted are similar to those used in the diagnosis of the subjects. The classification technique, which in this case, the MLP, is known to have good generalisation capabilities (Hertz, Krogh, Palmer, 1991). This contrasts the techniques used in previous attempts in the 1960's, e.g., the discriminant analysis, which is known to have poor generalisation capabilities. Thus, one of the reasons why this approach works may be attributed to the generalisation capabilities of the MLP.

## 5    Conclusions

In this paper, a method for classifying EEG data obtained from subjects who are normal, OCD or schizophrenia has been obtained by using the AR parameters as

input feature vectors. It is found that such a network has good generalisation capabilities.

## 6   Acknowledgments

The first and third author wish to acknowledge partial financial support from the Australian National Health and Medical Research Council. In addition, the first author wishes to acknowledge partial financial support from the Australian Research Council.

## 7   References

Basar, E. (1980). *EEG-Brain Dynamics – Relation between EEG and Brain Evoked Potentials.* Elsevier/North Holland Biomedical Press.

Cooper, R. (1980). *EEG Technology.* Butterworths. Third Editions.

Harris, F.J. (1978). " On the Use of windows for Harmonic Analysis with the Discrete Fourier Transform". *Proceedings IEEE.* Vol. 66, pp 51-83.

Herrmann, W.M. (1982). *Electroencephalography in Drug Research.* Butterworths.

Hertz, J. Krogh, A, Palmer, R. (1991) *Introduction to The Theory of Neural Computation.* Addison Wesley, Redwood City, Calif.

Kay, S.M., Marple, S.L., Jr. (1981). "Spectrum Analysis - A Modern Perspective". *Proceeding IEEE.* Vol. 69, No. 11, Nov. pp 1380 - 1417.

Kay, S.M. (1988) *Modern Spectral Estimation - Theory and Applications* Prentice hall.

Kolb, B., Whishaw, I.Q. (1990). *Fundamentals of Human Neuropsychology.* Freeman, New York.

Lindsay, D.F., Holmes, J.E. (1984). *Basic Human Neurophysiology.* Elsevier.

Lippmann, R.P. (1987) " An introduction to computing with neural nets" *IEEE Acoustics Speech and Signal Processing Magazine.* Vol. 4, No. 2, pp 4-22.

Lippmann, R.P., Moody, J., Touretzky, D.S. (Ed.) (1991). *Advances in Neural Information Processing Systems 3.* Morgan Kaufmann, San Mateo, Calif.

Marple, S.L., Jr. (1987). *Digital Spectral Analysis with Applications.* Prentice Hall.

Touretzky, D.S. (Ed.) (1989). *Advances in Neural Information Processing Systems 1.* Morgan Kaufmann, San Mateo, Calif.

Touretzky, D.S. (Ed.) (1990). *Advances in Neural Information Processing Systems 2.* Morgan Kaufmann, San Mateo, Calif.

## Footnotes

[1] From some preliminary work, it can be shown that this channel can be considered as a linear combination of the other channels, in the sense that the prediction error variance is small.

[2] It has been found that the EEG signal is approximately stationary for signal length of one second. Hence employing a data frame width of one second ensures that the underlying assumptions in the AR modelling technique are valid (Marple, 1988)

[3]The results shown are typical results. We have used different data files for training and testing. In most cases, the classification errors on the unseen data files are small, similar to those presented here.
